# Mixture Modeling by Affinity Propagation

**Brendan J. Frey and Delbert Dueck**
University of Toronto

Software and demonstrations available at www.psi.toronto.edu

## Abstract

Clustering is a fundamental problem in machine learning and has been approached in many ways. Two general and quite different approaches include iteratively fitting a mixture model (*e.g.*, using EM) and linking together pairs of training cases that have high affinity (*e.g.*, using spectral methods). Pair-wise clustering algorithms need not compute sufficient statistics and avoid poor solutions by directly placing similar examples in the same cluster. However, many applications require that each cluster of data be accurately described by a prototype or model, so affinity-based clustering – and its benefits – cannot be directly realized. We describe a technique called "affinity propagation", which combines the advantages of both approaches. The method learns a mixture model of the data by recursively propagating affinity messages. We demonstrate affinity propagation on the problems of clustering image patches for image segmentation and learning mixtures of gene expression models from microarray data. We find that affinity propagation obtains better solutions than mixtures of Gaussians, the $K$-medoids algorithm, spectral clustering and hierarchical clustering, and is both able to find a pre-specified number of clusters and is able to automatically determine the number of clusters. Interestingly, affinity propagation can be viewed as belief propagation in a graphical model that accounts for pairwise training case likelihood functions and the identification of cluster centers.

## 1 Introduction

Many machine learning tasks involve clustering data using a mixture model, so that the data in each cluster is accurately described by a probability model from a pre-defined, possibly parameterized, set of models [1]. For example, words can be grouped according to common usage across a reference set of documents, and segments of speech spectrograms can be grouped according to similar speaker and phonetic unit. As researchers increasingly confront more challenging and realistic problems, the appropriate class-conditional models become more sophisticated and much more difficult to optimize.

By marginalizing over hidden variables, we can still view many hierarchical learning problems as mixture modeling, but the class-conditional models become complicated and non-linear. While such class-conditional models may more accurately describe the problem at hand, the optimization of the mixture model often becomes much more difficult. Exact computation of the data likelihoods may not be feasible and exact computation of the sufficient statistics needed to update parameterized models may not be feasible. Further, the complexity of the model and the approximations used for the likelihoods and the sufficient statistics often produce an optimization surface with a large number of poor local minima.

A different approach to clustering ignores the notion of a class-conditional model, and

links together pairs of data points that have high affinity. The affinity or similarity (a real number in $[0, 1]$) between two training cases gives a direct indication of whether they should be in the same cluster. Hierarchical clustering and its Bayesian variants [2] is a popular affinity-based clustering technique, whereby a binary tree is constructed greedily from the leaves to the root, by recursively linking together pairs of training cases with high affinity. Another popular method uses a spectral decomposition of the *normalized* affinity matrix [4]. Viewing affinities as transition probabilities in a random walk on data points, modes of the affinity matrix correspond to clusters of points that are isolated in the walk [3,5].

We describe a new method that, for the first time to our knowledge, combines the advantages of model-based clustering and affinity-based clustering. Unlike previous techniques that construct and learn probability models of *transitions* between data points [6,7], our technique learns a probability model of the data itself. Like affinity-based clustering, our algorithm directly examines pairs of nearby training cases to help ascertain whether or not they should be in the same cluster. However, like model-based clustering, our technique uses a probability model that describes the data as a mixture of class-conditional distributions. Our method, called "affinity propagation", can be viewed as the sum-product algorithm or the max-product algorithm in a graphical model describing the mixture model.

## 2  A greedy algorithm: $K$-medoids

The first step in obtaining the benefit of pair-wise training case comparisons is to replace the parameters of the mixture model with pointers into the training data. A similar representation is used in $K$-medians clustering or $K$-medoids clustering, where the goal is to identify $K$ training cases, or *exemplars*, as cluster centers. Exact learning is known to be NP-hard (c.f. [8]), but a hard-decision algorithm can be used to find approximate solutions. While the algorithm makes greedy hard decisions for the cluster centers, it is a useful intermediate step in introducing affinity propagation.

For training cases $x_1, \ldots, x_N$, suppose the likelihood of training case $x_i$ given that training case $x_k$ is its cluster center is $P(x_i|x_i \text{ in } x_k)$ (*e.g.*, a Gaussian likelihood would have the form $e^{-(x_i-x_k)^2/2\sigma^2}/\sqrt{2\pi\sigma^2}$). Given the training data, this likelihood depends only on $i$ and $k$, so we denote it by $L_{ik}$. $L_{ii}$ is set to the Bayesian prior probability that $x_i$ is a cluster center. Initially, $K$ training cases are chosen as exemplars, *e.g.*, at random. Denote the current set of cluster center indices by $\mathcal{K}$ and the index of the current cluster center for $x_i$ by $s_i$. $K$-medoids iterates between assigning training cases to exemplars (E step), and choosing a training case as the new exemplar for each cluster (M step). Assuming for simplicity that the mixing proportions are equal and denoting the responsibility likelihood ratio by $r_{ik} = P(x_i|x_i \text{ in } x_k)/P(x_i|x_i \text{ not in } x_k)$[1], the updates are

E step
For $i = 1, \ldots, N$:
    For $k \in \mathcal{K}$: $r_{ik} \leftarrow L_{ik}/(\sum_{j:j\neq k} L_{ij})$
    $s_i \leftarrow \mathrm{argmax}_{k \in \mathcal{K}} \, r_{ik}$

Greedy M step
For $k \in \mathcal{K}$: Replace $k$ in $\mathcal{K}$ with $\mathrm{argmax}_{j:s_j=k} \left( \prod_{i:s_i=k} L_{ij} \right)$

This algorithm nicely replaces parameter-to-training case comparisons with pair-wise training case comparisons. However, in the greedy M step, specific training cases are chosen as exemplars. By not searching over all possible combinations of exemplars, the algorithm will frequently find poor local minima. We now introduce an algorithm that does approximately search over all possible combinations of exemplars.

# 3  Affinity propagation

The responsibilities in the greedy $K$-medoids algorithm can be viewed as messages that are sent from training cases to potential exemplars, providing soft evidence of the preference for each training case to be in each exemplar. To avoid making hard decisions for the cluster centers, we introduce messages called "availabilities". Availabilities are sent from exemplars to training cases and provide soft evidence of the preference for each exemplar to be available as a center for each training case.

Responsibilities are computed using likelihoods and availabilities, and availabilities are computed using responsibilities, recursively. We refer to both responsibilities and availabilities as affinities and we refer to the message-passing scheme as affinity propagation. Here, we explain the update rules; in the next section, we show that affinity propagation can be derived as the sum-product algorithm in a graphical model describing the mixture model. Denote the availability sent from candidate exemplar $x_k$ to training case $x_i$ by $a_{ki}$. Initially, these messages are set equal, $e.g.$, $a_{ki} = 1$ for all $i$ and $k$. Then, the affinity propagation update rules are recursively applied:

Responsibility updates

$$r_{ik} \leftarrow L_{ik} / (\sum_{j:j \neq k} a_{ij} L_{ij})$$

Availability updates

$$a_{kk} \leftarrow \prod_{j:j \neq k} (1 + r_{jk}) - 1$$
$$a_{ki} \leftarrow 1/(\frac{1}{r_{kk}} \prod_{j:j \neq k, j \neq i} (1 + r_{jk})^{-1} + 1 - \prod_{j:j \neq k, j \neq i} (1 + r_{jk})^{-1})$$

The first update rule is quite similar to the update used in EM, except the likelihoods used to normalize the responsibilities are modulated by the availabilities of the competing exemplars. In this rule, the responsibility of a training case $x_i$ as its own cluster center, $r_{ii}$, is high if no other exemplars are highly available to $x_i$ and if $x_i$ has high probability under the Bayesian prior, $L_{ii}$.

The second update rule also has an intuitive explanation. The availability of a training case $x_k$ as its own exemplar, $a_{kk}$, is high if at least one other training case places high responsibility on $x_k$ being an exemplar. The availability of $x_k$ as a exemplar for $x_i$, $a_{ki}$ is high if the self-responsibility $r_{kk}$ is high ($1/r_{kk} - 1$ approaches $-1$), but is decreased if other training cases compete in using $x_k$ as an exemplar (the term $1/r_{kk} - 1$ is scaled down if $r_{jk}$ is large for some other training case $x_j$).

Messages may be propagated in parallel or sequentially. In our implementation, each candidate exemplar absorbs and emits affinities in parallel, and the centers are ordered according to the sum of their likelihoods, $i.e.$ $\sum_i L_{ik}$. Direct implementation of the above propagation rules gives an $N^2$-time algorithm, but affinities need only be propagated between $i$ and $k$ if $L_{ik} > 0$. In practice, likelihoods below some threshold can be set to zero, leading to a sparse graph on which affinities are propagated.

Affinity propagation accounts for a Bayesian prior pdf on the exemplars and is able to automatically search over the appropriate number of exemplars. (Note that the number of exemplars is not pre-specified in the above updates.) In applications where a particular number of clusters is desired, the update rule for the responsibilities (in particular, the self-responsibilities $r_{kk}$, which determine the availabilities of the exemplars) can be modified, as described in the next section. Later, we describe applications where $K$ is pre-specified and where $K$ is automatically selected by affinity propagation.

The affinity propagation update rules can be derived as an instance of the sum-product

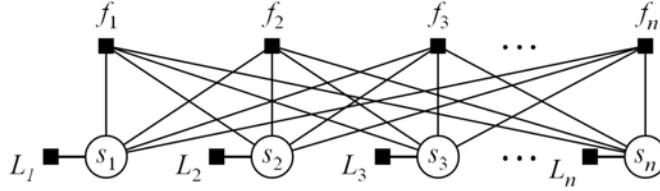

Figure 1: Affinity propagation can be viewed as belief propagation in this factor graph.

("loopy BP") algorithm in a graphical model. Using $s_i$ to denote the index of the exemplar for $x_i$, the product of the likelihoods of the training cases and the priors on the exemplars is $\prod_{i=1}^{N} L_{is_i}$. (If $s_i = i$, $x_i$ is an exemplar with *a priori* pdf $L_{ii}$.) The set of hidden variables $s_1, \ldots, s_N$ completely specifies the mixture model, but not all configurations of these variables are allowed: $s_i = k$ ($x_i$ in cluster $x_k$) implies $s_k = k$ ($x_k$ is an exemplar) and $s_k = k$ ($x_k$ is an exemplar) implies $s_i = k$ for some $i \neq k$ (some other training case is in cluster $x_k$). The global indicator function for the satisfaction of these constraints can be written $\prod_{k=1}^{N} f_k(s_1, \ldots, s_N)$, where $f_k$ is the constraint for candidate cluster $x_k$:

$$f_k(s_1, \ldots, s_N) = \begin{cases} 0 & \text{if } s_k = k \text{ and } s_i \neq k \text{ for all } i \neq k \\ 0 & \text{if } s_k \neq k \text{ and } s_i = k \text{ for some } i \neq k \\ 1 & \text{otherwise.} \end{cases}$$

Thus, the joint distribution of the mixture model and data factorizes as follows:

$$P = \prod_{i=1}^{N} L_{is_i} \prod_{k=1}^{N} f_k(s_1, \ldots, s_N).$$

The factor graph [10] in Fig. 1 describes this factorization. Each black box corresponds to a term in the factorization, and it is connected to the variables on which the term depends.

While exact inference in this factor graph is NP-hard, approximate inference algorithms can be used to infer the $s$ variables. It is straightforward to show that the updates for affinity propagation correspond to the message updates for the sum-product algorithm or loopy belief propagation (see [10] for a tutorial). The responsibilities correspond to messages sent from the $s$'s to the $f$'s, while the availabilities correspond to messages sent from the $f$'s to the $s$'s. If the goal is to find $K$ exemplars, an additional constraint $g(s_1, \ldots, s_N) = [K = \sum_{k=1}^{N} [s_k = k]]$ can be included, where [ ] indicates Iverson's notation ([true]=1 and [false] = 0). Messages can be propagated through this function in linear time, by implementing it as a Markov chain that accumulates exemplar counts.

**Max-product affinity propagation.** Max-product affinity propagation can be derived as an instance of the max-product algorithm, instead of the sum-product algorithm. The update equations for the affinities are modified and maximizations are used instead of summations. An advantage of max-product affinity propagation is that the algorithm is invariant to multiplicative constants in the *log-likelihoods*.

## 4  Image segmentation

A sensible model-based approach to image segmentation is to imagine that each patch in the image originates from one of a small number of prototype texture patches. The main difficulty is that in addition to standard additive or multiplicative pixel-level noise, another prevailing form of noise is due to transformations of the image features, and in particular translations.

Pair-wise affinity-based techniques and in particular spectral clustering has been employed with some success [4, 9], with the main disadvantage being that without an underlying

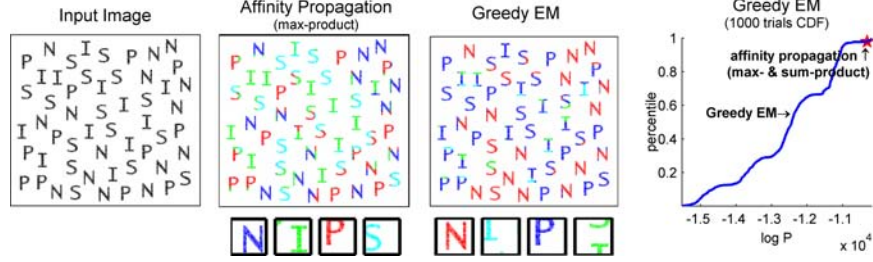

Figure 2: Segmentation of non-aligned gray-scale characters. Patches clustered by affinity propagation and $K$-medoids are colored according to classification (centers shown below solutions). Affinity propagation achieves a near-best score compared to 1000 runs of $K$-medoids.

model there is no sound basis for selecting good class representatives. Having a model with class representatives enables efficient synthesis (generation) of patches, and classification of test patches – requiring only $K$ comparisons (to class centers) rather than $N$ comparisons (to training cases).

We present results for segmenting two image types. First, as a toy example, we segment an image containing many noisy examples of the letters 'N' 'I' 'P' and 'S' (see Fig. 2). The original image is gray-scale with resolution $216 \times 240$ and intensities ranging from 0 (background color, white) to 1 (foreground color, black). Each training case $x_i$ is a $24 \times 24$ image patch and $x_i^m$ is the $m$th pixel in the patch. To account for translations, we include a hidden 2-D translation variable $T$. The match between patch $x_i$ and patch $x_k$ is measured by $\sum_m x_i^m \cdot f^m(x_k, T)$, where $f(x_k, T)$ is the patch obtained by applying a 2-D translation $T$ plus cropping to patch $x_k$. $f^m$ is the $m$th pixel in the translated, cropped patch. This metric is used in the likelihood function:

$$L_{ik} \propto \sum_T p(T) e^{\beta(\Sigma_m x_i^m \cdot f^m(x_k, T))/\bar{x}_i} \approx e^{\beta \max_T (\Sigma_m x_i^m \cdot f^m(x_k, T))/\bar{x}_i},$$

where $\bar{x}_i = \frac{1}{24^2} \sum_m x_i^m$ is used to normalize the match by the amount of ink in $x_i$. $\beta$ controls how strictly $x_i$ should match $x_k$ to have high likelihood. Max-product affinity propagation is independent of the choice of $\beta$, and for sum-product affinity propagation we quite arbitrarily chose $\beta = 1$. The exemplar priors $L_{kk}$ were set to $\mathrm{median}_{i,k \neq i} L_{ik}$.

We cut the image in Fig. 2 into a $9 \times 10$ grid of non-overlapping $24 \times 24$ patches, computed the pair-wise likelihoods, and clustered them into $K = 4$ classes using the greedy EM algorithm (randomly chosen initial exemplars) and affinity propagation. (Max-product and sum-product affinity propagation yielded identical results.) We then took a much larger set of overlapping patches, classified them into the 4 categories, and then colored each pixel in the image according to the most frequent class for the pixel. The results are shown in Fig. 2. While affinity propagation is deterministic, the EM algorithm depends on initialization. So, we ran the EM algorithm 1000 times and in Fig. 2 we plot the cumulative distribution of the $\log P$ scores obtained by EM. The score for affinity propagation is also shown, and achieves near-best performance (98$^{\text{th}}$ percentile).

We next analyzed the more natural $192 \times 192$ image shown in Fig. 3. Since there is no natural background color, we use mean-squared pixel differences in HSV color space to measure similarity between the $24 \times 24$ patches:

$$L_{ik} \propto e^{-\beta \min_T \Sigma_{m \in \mathcal{W}} (x_i^m - f^m(x_k, T))^2},$$

where $\mathcal{W}$ is the set of indices corresponding to a $16 \times 16$ window centered in the patch and $f^m(x_k, T)$ is the same as above. As before, we arbitrarily set $\beta = 1$ and $L_{kk}$ to $\mathrm{median}_{i,k \neq i} L_{ik}$.

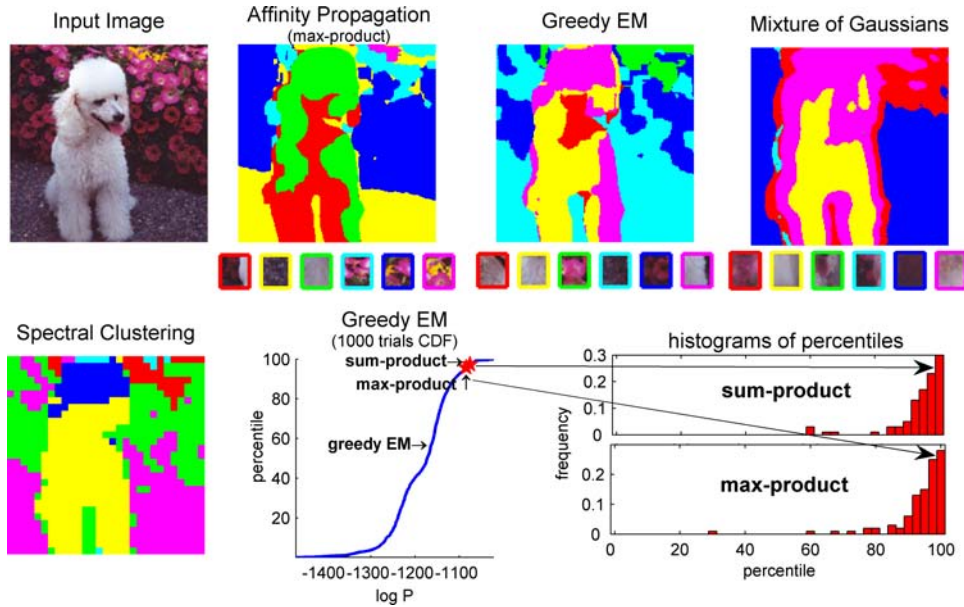

Figure 3: Segmentation results for several methods applied to a natural image. For methods other than affinity propagation, many parameter settings were tried and the best segmentation selected. The histograms show the percentile in score achieved by affinity propagation compared to 1000 runs of greedy EM, for different random training sets.

We cut the image in Fig. 3 into an $8 \times 8$ grid of non-overlapping $24 \times 24$ patches and clustered them into $K = 6$ classes using affinity propagation (both forms), greedy EM in our model, spectral clustering (using a normalized $L$-matrix based on a set of $29 \times 29$ overlapping patches), and mixtures of Gaussians[2]. For greedy EM, the affinity propagation algorithms, and mixtures of Gaussians, we then choose all possible $24 \times 24$ overlapping patches and calculated the likelihoods of them given each of the 6 cluster centers, classifying each patch according to its maximum likelihood.

Fig. 3 shows the segmentations for the various methods, where the central pixel of each patch is colored according to its class. Again, affinity propagation achieves a solution that is near-best compared to one thousand runs of greedy EM.

## 5    Learning mixtures of gene models

Currently, an important problem in genomics research is the discovery of genes and gene variants that are expressed as messenger RNAs (mRNAs) in normal tissues. In a recent study [11], we used DNA-based techniques to identify 837,251 possible exons ("putative exons") in the mouse genome. For each putative exon, we used an Agilent microarray probe to measure the amount of corresponding mRNA that was present in each of 12 mouse tissues. Each 12-D vector, called an "expression profile", can be viewed as a feature vector indicating the putative exon's function. By grouping together feature vectors for nearby probes, we can detect genes and variations of genes. Here, we compare affinity propagation with hierarchical clustering, which was previously used to find gene structures [12].

Fig. 4a shows a normalized subset of the data and gives three examples of groups of nearby

(a)

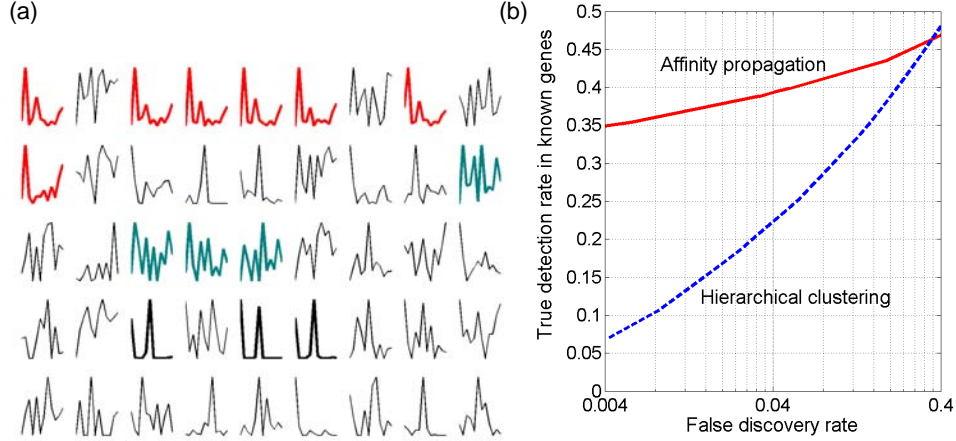

(b)

Figure 4: (a) A normalized subset of 837,251 tissue expression profiles – mRNA level versus tissue – for putative exons from the mouse genome (most profiles are much noisier than these). (b) The true exon detection rate (in known genes) versus the false discovery rate, for affinity propagation and hierarchical clustering.

feature vectors that are similar enough to provide evidence of gene units. The actual data is generally much noisier, and includes multiplicative noise (exon probe sensitivity can vary by two orders of magnitude), correlated additive noise (a probe can cross-hybridize in a tissue-independent manner to background mRNA sources), and spurious additive noise (due to a noisy measurement procedure and biological effects such as alternative splicing). To account for noise, false putative exons, and the distance between exons in the same gene, we used the following likelihood function:

$$L_{ij} = \lambda e^{-\lambda|i-j|} \left( q \cdot p_0(x_i) + (1-q) \int p(y,z,\sigma) \frac{e^{-\frac{1}{2\sigma^2} \Sigma_{m=1}^{12} (x_i^m - (y \cdot x_j^m + z))^2}}{\sqrt{2\pi\sigma^2}^{12}} dy\,dz\,d\sigma \right)$$

$$\approx \lambda e^{-\lambda|i-j|} \left( q \cdot p_0(x_i) + (1-q) \max_{y,z,\sigma} p(y,z,\sigma) \frac{e^{-\frac{1}{2\sigma^2} \Sigma_{m=1}^{12} (x_i^m - (y \cdot x_j^m + z))^2}}{\sqrt{2\pi\sigma^2}^{12}} \right),$$

where $x_i^m$ is the expression level for the $m$th tissue in the $i$th probe (in genomic order). We found that in this application, the maximum is a sufficiently good approximation to the integral. The distribution over the distance between probes in the same gene $|i - j|$ is assumed to be geometric with parameter $\lambda$. $p_0(x_i)$ is a background distribution that accounts for false putative exons and $q$ is the probability of a false putative exon within a gene. We assumed $y$, $z$ and $\sigma$ are independent and uniformly distributed[3]. The Bayesian prior probability that $x_k$ is an exemplar is set to $\theta \cdot p_0(x_k)$, where $\theta$ is a control knob used to vary the sensitivity of the system.

Because of the term $\lambda e^{-\lambda|i-j|}$ and the additional assumption that genes on the same strand do not overlap, it is not necessary to propagate affinities between all $837,251^2$ pairs of training cases. We assume $L_{ij} = 0$ for $|i - j| > 100$, in which case it is not necessary to propagate affinities between $x_i$ and $x_j$. The assumption that genes do not overlap implies that if $s_i = k$, then $s_j = k$ for $j \in \{\min(i,k), \ldots, \max(i,k)\}$. It turns out that this constraint causes the dependence structure in the update equations for the affinities to reduce to a chain, so affinities need only be propagated forward and backward along the genome. After affinity propagation is used to automatically select the number of mixture

components and identify the mixture centers and the probes that belong to them (genes), each probe $x_i$ is labeled as an exon or a non-exon depending on which of the two terms in the above likelihood function ($q \cdot p_0(x_i)$ or the large term to its right) is larger.

Fig. 4b shows the fraction of exons in known genes detected by affinity propagation versus the false detection rate. The curve is obtained by varying the sensitivity parameter, $\theta$. The false detection rate was estimated by randomly permuting the order of the probes in the training set, and applying affinity propagation. Even for quite low false discovery rates, affinity propagation identifies over one third of the known exons. Using a variety of metrics, including the above metric, we also used hierarchical clustering to detect exons. The performance of hierarchical clustering using the metric with highest sensitivity is also shown. Affinity propagation has significantly higher sensitivity, *e.g.*, achieving a five-fold increase in true detection rate at a false detection rate of 0.4%.

## 6 Computational efficiency

The following table compares the MATLAB execution times of our implementations of the methods we compared on the problems we studied. For methods that first compute a likelihood or affinity matrix, we give the timing of this computation first. Techniques denoted by "*" were run many times to obtain the shown results, but the given time is for a single run.

|  | Affinity Prop | $K$-medoids* | Spec Clust* | MOG EM* | Hierarch Clust |
|---|---|---|---|---|---|
| NIPS | 12.9 s + 2.0 s | 12.9 s + .2 s | - | - | - |
| Dog | 12.0 s + 1.5 s | 12.0 s + 0.1 s | 12.0 s + 29 s | 3.3 s | - |
| Genes | 16 m + 43 m | - | - | - | 16 m + 28 m |

## 7 Summary

An advantage of affinity propagation is that the update rules are deterministic, quite simple, and can be derived as an instance of the sum-product algorithm in a factor graph. Using challenging applications, we showed that affinity propagation obtains better solutions (in terms of percentile log-likelihood, visual quality of image segmentation and sensitivity-to-specificity) than other techniques, including $K$-medoids, spectral clustering, Gaussian mixture modeling and hierarchical clustering.

To our knowledge, affinity propagation is the first algorithm to combine advantages of pair-wise clustering methods that make use of bottom-up evidence and model-based methods that seek to fit top-down global models to the data.

## Footnotes

[1]Note that using the traditional definition of responsibility, $r_{ik} \leftarrow L_{ik}/(\Sigma_j L_{ij})$, will give the same decisions as using the likelihood ratio.

[2]For spectral clustering, we tried $\beta = 0.5$, 1 and 2, and for each of these tried clustering using 6, 8, 10, 12 and 14 eigenvectors. We then visually picked the best segmentation ($\beta = 1$, 10 eigenvectors). The eigenvector features were clustered using EM in a mixture of Gaussians and out of 10 trials, the solution with highest likelihood was selected. For mixtures of Gaussians applied directly to the image patches, we picked the model with highest likelihood in 10 trials.

[3]Based on the experimental procedure and a set of previously-annotated genes (RefSeq), we estimated $\lambda = 0.05$, $q = 0.7$, $y \in [.025, 40]$, $z \in [-\mu, \mu]$ (where $\mu = \max_{i,m} x_i^m$), $\sigma \in (0, \mu]$. We used a mixture of Gaussians for $p_0(x_i)$, which was learned from the entire training set.

## References

[1] CM Bishop. *Neural Networks for Pattern Recognition*. Oxford University Press, NY, 1995.

[2] KA Heller, Z Ghahramani. Bayesian hierarchical clustering. *ICML*, 2005.

[3] M Meila, J Shi. Learning segmentation by random walks. *NIPS 14*, 2001.

[4] J Shi, J Malik. Normalized cuts and image segmentation. *Proc CVPR*, 731-737, 1997.

[5] A Ng, M Jordan, Y Weiss. On spectral clustering: Analysis and an algorithm. *NIPS 14*, 2001.

[6] N Shental A Zomet T Hertz Y Weiss. Pairwise clustering and graphical models *NIPS 16* 2003.

[7] R Rosales, BJ Frey. Learning generative models of affinity matrices. *Proc UAI*, 2003.

[8] M Charikar, S Guha, A Tardos, DB Shmoys. A constant-factor approximation algorithm for the $k$-median problem. *J Comp and Sys Sci*, **65:1**, 129-149, 2002.

[9] J Malik *et al.*. Contour and texture analysis for image segmentation. *IJCV* **43:1**, 2001.

[10] FR Kschischang, BJ Frey, H-A Loeliger. Factor graphs and the sum-product algorithm. *IEEE Trans Info Theory* **47:2**, 498-519, 2001.

[11] BJ Frey, QD Morris, M Robinson, TR Hughes. Finding novel transcripts in high-resolution genome-wide microarray data using the GenRate model. *Proc RECOMB 2005*, 2005.

[12] D. D. Shoemaker *et al.* Experimental annotation of the human genome using microarray technology. *Nature* **409**, 922-927, 2001.
